# Learning in the Vestibular System: Simulations of Vestibular Compensation Using Recurrent Back-Propagation

**Thomas J. Anastasio**
University of Illinois
Beckman Institute
405 N. Mathews Ave.
Urbana, IL 61801

## Abstract

Vestibular compensation is the process whereby normal functioning is regained following destruction of one member of the pair of peripheral vestibular receptors. Compensation was simulated by lesioning a dynamic neural network model of the vestibulo-ocular reflex (VOR) and retraining it using recurrent back-propagation. The model reproduced the pattern of VOR neuron activity experimentally observed in compensated animals, but only if connections heretofore considered uninvolved were allowed to be plastic. Because the model incorporated nonlinear units, it was able to reconcile previously conflicting, linear analyses of experimental results on the dynamic properties of VOR neurons in normal and compensated animals.

## 1 VESTIBULAR COMPENSATION

Vestibular compensation is one of the oldest and most well studied paradigms in motor learning. Although it is neurophysiologically well described, the adaptive mechanisms underlying vestibular compensation, and its effects on the dynamics of vestibular responses, are still poorly understood. The purpose of this study is to gain insight into the compensatory process by simulating it as learning in a recurrent neural network model of the vestibulo-ocular reflex (VOR).

The VOR stabilizes gaze by producing eye rotations that counterbalance head rotations. It is mediated by brainstem neurons in the vestibular nuclei (VN) that relay head velocity signals from vestibular sensory afferent neurons to the motoneurons of the eye muscles (Wilson and Melvill Jones 1979). The VOR circuitry also processes the canal signals, stretching out their time constants by four times before transmitting this signal to the motoneurons. This process of time constant lengthening is known as velocity storage (Raphan et al. 1979).

The VOR is a bilaterally symmetric structure that operates in push-pull. The VN are linked bilaterally by inhibitory commissural connections. Removal of the vestibular receptors from one side (hemilabyrinthectomy) unbalances the system, resulting in continuous eye movement that occurs in the absence of head movement, a condition known as spontaneous nystagmus. Such a lesion also reduces VOR sensitivity (gain) and eliminates velocity storage. Compensatory restoration of VOR occurs in stages (Fetter and Zee 1988). It begins by quickly eliminating spontaneous nystagmus, and continues by increasing VOR gain. Curiously, velocity storage never recovers.

## 2 NETWORK ARCHITECTURE

The horizontal VOR is modeled as a three-layered neural network (Figure 1). All of the units are nonlinear, passing their weighted input sums through the sigmoidal squashing function. This function bounds unit responses between zero and one. Input units represent afferents from the left (*lhc*) and right (*rhc*) horizontal semicircular canal receptors. Output units correspond to motoneurons of the lateral (*lr*) and medial (*mr*) rectus muscles of the left eye. Interneurons in the VN are represented by hidden units on the left (*lvn1*, *lvn2*) and right (*rvn1*, *rvn2*) sides of the model brainstem. Bias units stand for non-vestibular inputs, on the left (*lb*) and right (*rb*) sides.

Network connectivity reflects the known anatomy of mammalian VOR (Wilson and Melvill Jones 1979). Vestibular commissures are modeled as recurrent connections between hidden units on opposite sides. All connection weights to the hidden units are plastic, but those to the outputs are initially fixed, because it is generally believed that synaptic plasticity occurs only at the VN level in vestibular compensation (Galiana et al. 1984). Fixed hidden-to-output weights have a crossed, reciprocal pattern.

## 3 TRAINING THE NORMAL NETWORK

The simulations began by training the network shown in Figure 1, with both vestibular inputs intact (normal network), to produce the VOR with velocity storage (Anastasio 1991). The network was trained using recurrent back-propagation (Williams and Zipser 1989). The input and desired output sequences correspond to the canal afferent signals and motoneuron eye-velocity commands that would produce the VOR response to two impulse head rotational accelerations, one to the left and the other to the right. One input (*rhc*) and desired output (*lr*) sequence is shown in Figure 2A (dotted and dashed, respectivley). Those for *lhc* and *mr* (not shown) are identical but inverted. The desired output responses are equal in amplitude to the inputs, producing VOR

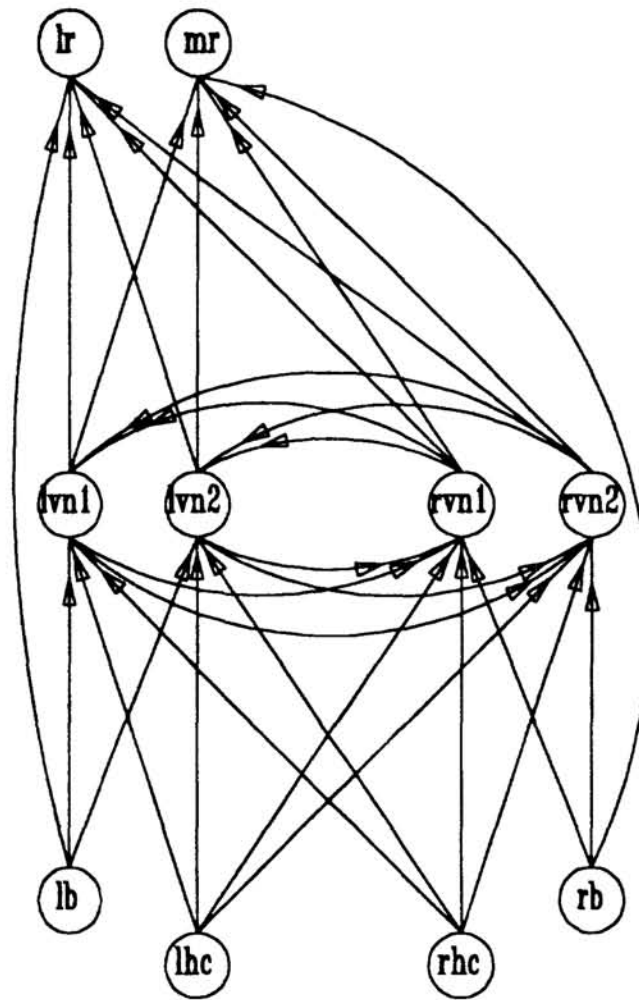

Figure 1.    Recurrent Neural Network Model of the Horizontal Vestibulo-Ocular Reflex (VOR).  *lhc, rhc*: left and right horizontal semicircular canal afferents; *lvn1, lvn2, rvn1, rvn2*: vestibular nucleus neurons on left and right sides of model brainstem; *lr, mr*: lateral and medial rectus muscles of left eye; *lb, rb*: left and right non-vestibular inputs.  This and subsequent figures redrawn from Anastasio (in press).

eye movements that would perfectly counterbalance head movements.  The output responses decay more slowly than the input responses, reflecting velocity storage. Between head movements, both desired outputs have the same spontaneous firing rate of 0.50.  With output spontaneous rates (SRs) balanced, no push-pull eye velocity command is given and, consequently, no VOR eye movement would be made.

The normal network learns the VOR transformation after about 4,000 training sequence presentations (passes).  The network develops reciprocal connections from input to hidden units, as in the actual VOR (Wilson and Melvill Jones 1979). Inhibitory recurrent connections form an integrating (*lvn1*, *rvn1*) and a non-integrating (*lvn2*, *rvn2*) pair of hidden units (Anastasio 1991).   The integrating pair subserve storage in the network. They have strong  mutual inhibition and exert net positive feedback on themselves. The non-integrating pair have almost no mutual inhibition.

## 4 SIMULATING VESTIBULAR COMPENSATION

After the normal network is constructed, with both inputs intact, vestibular compensation can be simulated by removing the input from one side and retraining with recurrent back-propagation. Left hemilabyrinthectomy produces deficits in the model that correspond to those observed experimentally. The responses of output unit *lr* acutely (i.e. immediately) following left input removal are shown in Figure 2A. The SR of *lr* (solid) is greatly increased above normal (dashed); that of *mr* (not shown) is decreased by the same amount. This output SR imbalance would result in eye movement to the left in the absence of head movement (spontaneous nystagmus). The gain of the outputs is greatly decreased. This is due to removal of one half the network input, and to the SR imbalance forcing the output units into the low gain extremes of the squashing function. Velocity storage is also eliminated by left input removal, due to events at the hidden unit level (see below).

During retraining, the time course of simulated compensation is similar to that

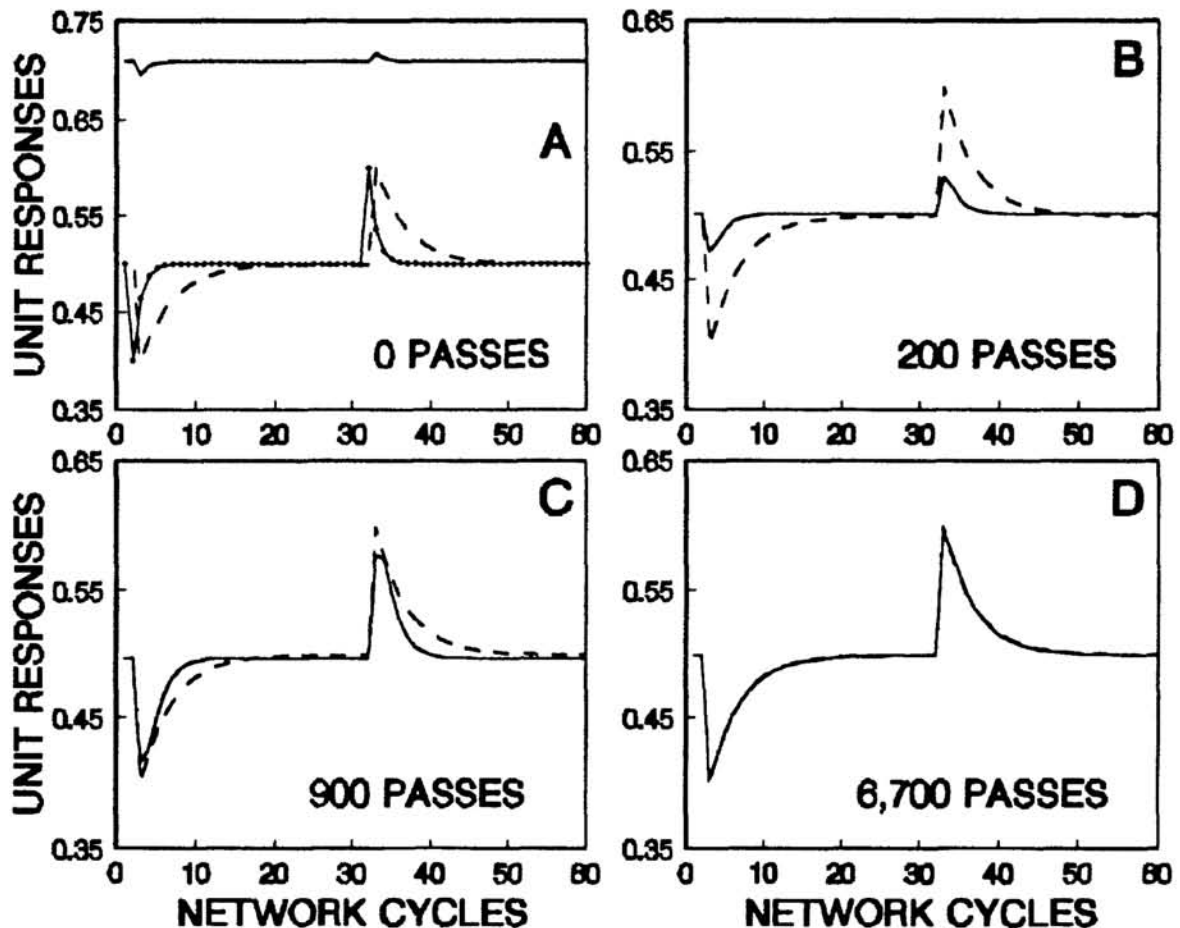

Figure 2. Simulated Compensation in the VOR Neural Network Model. Response of *lr* (solid) is shown at each stage of compensation: A, acutely (i.e. immediately) following the lesion; B, after spontaneous nystagmus has been eliminated; C, after VOR gain has been largely restored; D, after full recovery of VOR. Desired response of *lr* (dashed) shown in all plots. Intact input from *rhc* (dotted) shown in A only.

observed experimentally (Fetter and Zee 1988). Spontaneous nystagmus is eliminated after 200 passes, as the SRs of the output units are brought back to their normal level (Figure 2B). Output unit gain is largely restored by 900 passes, but time constant remains close to that of the inputs (Figure 2C). At this stage, VOR gain would have increased substantially, but its time constant would remain low, indicating loss of velocity storage. This stage approximates the extent of experimentally observed compensation (ibid.). Completely restoring the normal VOR, with full velocity storage, requires over seven times more retraining (Figure 2D).

The responses of the hidden units during each stage of simulated compensation are shown in Figure 3A and 3C. Average hidden unit SR and gain are shown as dotted lines in Figure 3A and 3C, respectively. Acutely following left input removal (AC stage), the SRs of left (dashed) and right (solid) hidden units decrease and increase, respectively (Figure 3A). One left hidden unit (*lvn1*) is actually silenced. Hidden unit gain at AC stage is greatly reduced bilaterally (Figure 3C), as for the outputs.

At the point where spontaneous nystagmus is eliminated (NE stage), hidden units SRs are balanced bilaterally, and none of the units are spontaneously silent (Figure 3A). When VOR gain is largely restored (GR stage, corresponding to experimentally observed compensation), the gains of the hidden units have substantially increased (Figure 3C). At GR stage, average hidden unit SR has also increased but the bilateral SR balance has been strictly maintained (Figure 3A). A comparison with experimental data (Yagi and Markham 1984; Newlands and Perachio 1990) reveals that the behavior of hidden units in the model does *not* correspond to that observed for real VN neurons in compensated animals. Rather than having bilateral SR balance, the average SR of VN neurons in compensated animals is lower on the lesion-side and higher on the intact-side. Moreover, many lesion-side VN neurons are permanently silenced. Also, rather than substantially recovering gain, the gains of VN neurons in compensated animals increase little from their low values acutely following the lesion.

The network model adopts its particular (and unphysiological) solution to vestibular compensation because, with fixed connection weights to the outputs, compensation can be brought about only by changes in hidden unit behavior. Thus, output SRs will be balanced only if hidden SRs are balanced, and output gain will increase only if hidden gain increases. The discrepancy between model and actual VN neuron data suggests that compensation cannot rely solely on synaptic plasticity at the VN level.

# 5 RELAXING CONSTRAINTS

A better match between model and experimental VN neuron data can be achieved by rerunning the compensation simulation with modifiable weights at all allowed network connections (Figure 1). Bias-to-output and hidden-to-output synaptic weights, which were previously fixed, are now made plastic. These extra degrees of freedom give the adapting network greater flexibility in achieving compensation, and release it from a strict dependency upon the behavior of the hidden units. The time course of compensation in the all-weights-modifiable example is similar to the previous case (Figure 2), but each stage is reached after fewer passes.

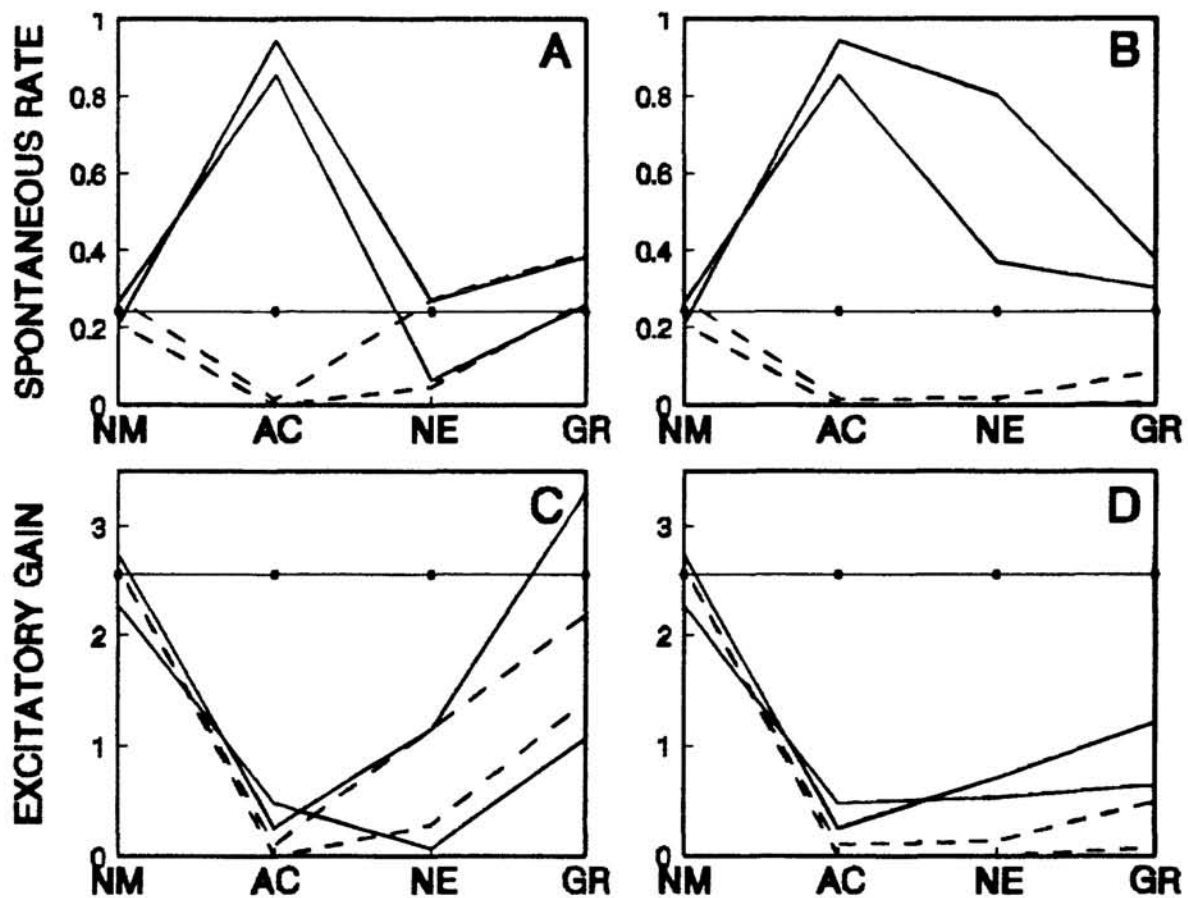

Figure 3.  Behavior of Hidden Units at Various Stages of Compensation in the VOR Neural Network Model.  Spontaneous rate (SR, A and B) and gain (C and D) are shown for networks with hidden layer weights only modifiable (A and C) or with all weights modifiable (B and D).  Normal average SR (A and B) and gain (C and D) shown as dotted lines.  NM, normal stage; AC, acutely following lesion; NE, after spontaneous nystagmus is eliminated; GR, after VOR gain is largely restored.

The behavior of the hidden units in the all-weights simulation more closely matches that of actual VN neurons in compensated animals (Figure 3B and 3D).  At NE stage, even though spontaneous nystagmus is eliminated, there remains a large bilateral imbalance in hidden unit SR, and one lesion-side hidden unit (*lvn1*) is silenced (Figure 3B).  At GR stage, hidden unit gain has increased only modestly from the low acute level (Figure 3D), and the bilateral SR imbalance persists, with *lvn1* still essentially spontaneously silent (Figure 3B).  This modeling result constitutes a testable prediction that synaptic plasticity is occurring at the motoneuron as well as at the VN level in vestibular compensation.

# 6 NETWORK DYNAMICS

In the all-weights simulation at GR stage, as well as in compensated animals, some lesion-side VN neurons are silenced.  Hidden unit *lvn1* is silenced by its inhibitory commissural interaction with *rvn1*, which in the normal network allowed the pair to

form an integrating, recurrent loop. Silencing of *lvn1* breaks the commissural loop and consequently eliminates velocity storage in the network. VN neuron silencing could also account for the loss of velocity storage in the real, compensated VOR.

Loss of velocity storage in the model, in response to step head rotational acceleration stimuli, is shown in Figure 4. The output step response that would be expected given the longer VOR time constant is shown for *lr* in Figure 4A (dashed). The response of *mr* (not shown) is identical but inverted. Instead of expressing the longer VOR time constant, the actual step response of *lr* in the all-weights compensated network at GR stage (Figure 4A, dotted) has a rise time constant that is equal to the canal time constant, indicating complete loss of velocity storage. This is due to the behavior of the hidden units. The step responses of the integrating pair of hidden units in the compensated network at GR stage are shown in Figure 4B (*lvn1*, lower dotted; *rvn1*, upper dotted). Velocity storage is eliminated because *lvn1* is silenced, and this breaks the commissural loop that supports integration in the network.

Paradoxically, in the normal network with all hidden units spontaneously active, the output step response rise time constant is also equal to that of the canal afferents, again indicating a loss of velocity storage. This is shown for *lr* from the normal network in Figure 4A (solid). The step responses of the hidden units in the normal network are shown in Figure 4B (*lvn1*, dashed; *rvn1*, solid). Unit *lvn1*, which is spontaneously active in the normal network, is quickly driven into cut-off by the step stimulus. This breaks the commissural loop and eliminates velocity storage, accounting for the short rise time constants of hidden and output units network wide.

This result can explain some conflicting experimental findings concerning the

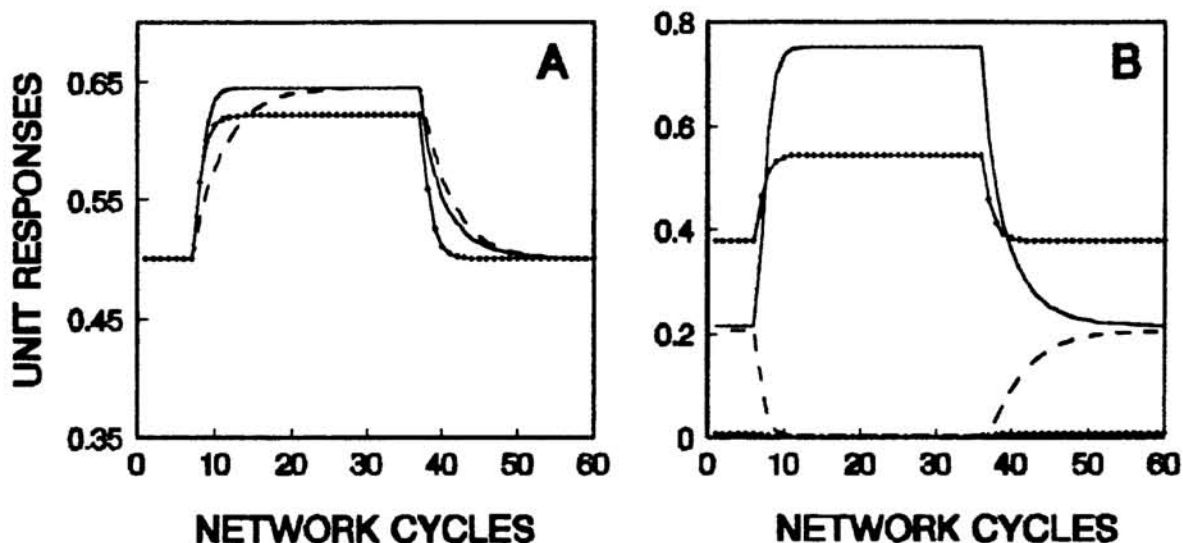

Figure 4. Responses of Units to Step Head Rotational Acceleration Stimuli in VOR Neural Network Model. A, expected response of *lr* with VOR time constant (dashed), and actual responses of *lr* in normal (solid) and all-weights compensated (dotted) networks. B, response of *lvn1* (dashed) and *rvn1* (solid) in normal network, and of *lvn1* (lower dotted) and *rvn1* (upper dotted) in all-weights compensated network.

dynamics of VN neurons in normal and compensated animals. Using sinusoidal stimuli, the time constants of VN neurons were found to be lower in compensated than in normal gerbils (Newlands and Perachio 1990). In contrast, using step stimuli, no difference in rise time constants were found for VN neurons in normal as compared to compensated cats (Yagi and Markham 1984).

Rather than being a species difference, the disagreement may involve the type of stimulus used. Step accelerations are intense stimuli that can drive VN neurons to extreme levels. In response to a step in their off-directions, many VN neurons in normal cats were observed to cut-off (ibid.). As shown in Figure 4, this would disrupt commissural interactions and reduce velocity storage and VN neuron rise time constants, just as if these neurons were silenced as they are in compensated animals. In fact, VN neuron rise time constants were observed to be low in both normal and compensated cats (ibid.). In contrast, sinusoidal stimuli at an intensity that does not cause widespread VN neuron cut-off would not be expected to disrupt velocity storage in normal animals.

## Acknowledgements

This work was supported by a grant from the Whitaker Foundation.

## References

Anastasio TJ (1991) Neural network models of velocity storage in the horizontal vestibulo-ocular reflex. Biol Cybern 64:187-196

Anastasio TJ (in press) Simulating vestibular compensation using recurrent back-propagation. Biol Cybern

Fetter M, Zee DS (1988) Recovery from unilateral labyrinthectomy in rhesus monkey. J Neurophysiol 59:370-393

Galiana HL, Flohr H, Melvill Jones G (1984) A reevauation of intervestibular nuclear coupling: its role in vestibular compensation. J Neurophysiol 51:242-259

Newlands SD, Perachio AA (1990) Compensation of horizontal canal related activity in the medial vestibular nucleus following unilateral labyrinth ablation in the decerebrate gerbil. I. type I neurons. Exp Brain Res 82:359-372

Raphan Th, Matsuo V, Cohen B (1979) Velocity storage in the vestibulo-ocular reflex arc (VOR). Exp Brain Res 35:229-248

Williams RJ, Zipser D (1989) A learning algorithm for continually running fully recurrent neural networks. Neural Comp 1:270-280

Wilson VJ, Melvill Jones G (1979) Mammalian vestibular physiology. Plenum Press, New York

Yagi T, Markham CH (1984) Neural correlates of compensation after hemilabyrinthectomy. Exp Neurol 84:98-108
